# Gradient Descent for General Reinforcement Learning

**Leemon Baird**
leemon@cs.cmu.edu
www.cs.cmu.edu/~leemon
Computer Science Department
5000 Forbes Avenue
Carnegie Mellon University
Pittsburgh, PA 15213-3891

**Andrew Moore**
awm@cs.cmu.edu
www.cs.cmu.edu/~awm
Computer Science Department
5000 Forbes Avenue
Carnegie Mellon University
Pittsburgh, PA 15213-3891

## Abstract

A simple learning rule is derived, the *VAPS* algorithm, which can be instantiated to generate a wide range of new reinforcement-learning algorithms. These algorithms solve a number of open problems, define several new approaches to reinforcement learning, and unify different approaches to reinforcement learning under a single theory. These algorithms all have guaranteed convergence, and include modifications of several existing algorithms that were known to fail to converge on simple MDPs. These include $Q$-learning, SARSA, and advantage learning. In addition to these *value-based* algorithms it also generates pure *policy-search* reinforcement-learning algorithms, which learn optimal policies without learning a value function. In addition, it allows policy-search and value-based algorithms to be combined, thus unifying two very different approaches to reinforcement learning into a single Value and Policy Search (VAPS) algorithm. And these algorithms converge for POMDPs without requiring a proper belief state. Simulations results are given, and several areas for future research are discussed.

## 1 CONVERGENCE OF GREEDY EXPLORATION

Many reinforcement-learning algorithms are known that use a parameterized function approximator to represent a value function, and adjust the weights incrementally during learning. Examples include $Q$-learning, SARSA, and advantage learning. There are simple MDPs where the original form of these algorithms fails to converge, as summarized in Table 1. For the cases with √, the algorithms are guaranteed to converge under reasonable assumptions such as

Table 1. Current convergence results for incremental, value-based RL algorithms. Residual algorithms changed every **X** in the first two columns to √. The new algorithms proposed in this paper change every **X** to a √.

| | | Fixed distribution (on-policy) | Fixed distribution | Usually-greedy distribution |
|---|---|---|---|---|
| Markov chain | Lookup table | √ | √ | |
| | Averager | √ | √ | |
| | Linear | √ | **X** | |
| | Nonlinear | **X** | **X** | |
| MDP | Lookup table | √ | √ | √ |
| | Averager | √ | √ | **X** |
| | Linear | **X** | **X** | **X** |
| | Nonlinear | **X** | **X** | **X** |
| POMDP | Lookup table | √ | √ | **X** |
| | Averager | √ | √ | **X** |
| | Linear | **X** | **X** | **X** |
| | Nonlinear | **X** | **X** | **X** |

√=convergence guaranteed
**X**=counterexample is known that either diverges or oscillates between the best and worst possible policies.

decaying learning rates. For the cases with **X**, there are known counterexamples where it will either diverge or oscillate between the best and worst possible policies, which have very-different values. This can happen even with infinite training time and slowly-decreasing learning rates (Baird, 95, Gordon, 96). Each **X** in the first two columns can be changed to a √ and made to converge by using a modified form of the algorithm, the *residual* form (Baird 95). But this is only possible when learning with a fixed training distribution, and that is rarely practical. For most large problems, it is useful to explore with a policy that is usually-greedy with respect to the current value function, and that changes as the value function changes. In that case (the rightmost column of the chart), the current convergence guarantees are not very good. One way to guarantee convergence in all three columns is to modify the algorithm so that it is performing stochastic gradient descent on some average error function, where the average is weighted by state-visitation frequencies for the current usually-greedy policy. Then the weighting changes as the policy changes. It might appear that this gradient is difficult to compute. Consider $Q$-learning exploring with a Boltzman distribution that is usually greedy with respect to the learned $Q$ function. It seems difficult to calculate gradients, since changing a single weight will change many $Q$ values, changing a single $Q$ value will change many action-choice probabilities in that state, and changing a single action-choice probability may affect the frequency with which every state in the MDP is visited. Although this might seem difficult, it is not. Surprisingly, unbiased estimates of the gradients of visitation distributions with respect to the weights can be calculated quickly, and the resulting algorithms can put a √ in every case in Table 1.

## 2 DERIVATION OF THE VAPS EQUATION

Consider a sequence of transitions observed while following a particular stochastic policy on an MDP. Let $s_t = \{x_0, u_0, R_0, \ x_1, u_1, R_1, \ \ldots \ x_{t-1}, u_{t-1}, R_{t-1}, \ x_t, u_t, R_t\}$ be the sequence of states, actions, and reinforcements up to time $t$, where performing action $u_t$ in state $x_t$ yields reinforcement $R_t$ and a transition to state $x_{t+1}$. The

stochastic policy may be a function of a vector of weights **w**. Assume the MDP has a single start state named $x_0$. If the MDP has terminal states, and $x_t$ is a terminal state, then $x_{t+1}=x_0$. Let $\mathbf{S}_t$ be the set of all possible sequences from time 0 to $t$. Let $e(s_t)$ be a given error function that calculates an error on each time step, such as the squared Bellman residual at time $t$, or some other error occurring at time $t$. If $e$ is a function of the weights, then it must be a smooth function of the weights. Consider a period of time starting at time 0 and ending with probability $P(\text{end}|s_t)$ after the sequence $s_t$ occurs. The probabilities must be such that the expected squared period length is finite. Let $B$ be the expected total error during that period, where the expectation is weighted according to the state-visitation frequencies generated by the given policy:

$$B = \sum_{T=0}^{\infty} \sum_{s_T \in \mathbf{S}_T} P(\text{period ends at time } T \text{ after trajectory } s_T) \sum_{t=0}^{T} e(s_t) \quad (1)$$

$$= \sum_{t=0}^{\infty} \sum_{s_t \in \mathbf{S}_t} e(s_t) P(s_t) \quad (2)$$

where:

$$P(s_t) = P(u_t \mid s_t) P(R_t \mid s_t) \prod_{i=0}^{t-1} P(u_t \mid s_t) P(R_t \mid s_t) P(s_{t+1} \mid \mathbf{s}_t) [1 - P(\text{end} \mid s_t)] \quad (3)$$

Note that on the first line, for a particular $s_t$, the error $e(s_t)$ will be added in to $B$ once for every sequence that starts with $s_t$. Each of these terms will be weighted by the probability of a complete trajectory that starts with $s_t$. The sum of the probabilities of all trajectories that start with $s_t$ is simply the probability of $s_t$ being observed, since the period is assumed to end eventually with probability one. So the second line equals the first. The third line is the probability of the sequence, of which only the $P(u_t|x_t)$ factor might be a function of **w**. If so, this probability must be a smooth function of the weights and nonzero everywhere. The partial derivative of $B$ with respect to $w$, a particular element of the weight vector **w**, is:

$$\frac{\partial}{\partial w} B = \sum_{t=0}^{\infty} \sum_{s_t \in \mathbf{S}_t} \left( \left( \frac{\partial}{\partial w} e(s_t) \right) P(s_t) + e(s_t) P(s_t) \sum_{j=1}^{t} \frac{\frac{\partial}{\partial w}\left[ P(u_{j-1} \mid s_{j-1}) \right]}{P(u_{j-1} \mid s_{j-1})} \right) \quad (4)$$

$$= \sum_{t=0}^{\infty} \sum_{s_t \in \mathbf{S}_t} P(s_t) \left[ \frac{\partial}{\partial w} e(s_t) + e(s_t) \sum_{j=1}^{t} \frac{\partial}{\partial w} \ln\left( P(u_{j-1} \mid s_{j-1}) \right) \right] \quad (5)$$

Space here is limited, and it may not be clear from the short sketch of this derivation, but summing (5) over an entire period does give an unbiased estimate of $B$, the expected total error during a period. An incremental algorithm to perform stochastic gradient descent on $B$ is the weight update given on the left side of Table 2, where the summation over previous time steps is replaced with a trace $T_t$ for each weight. This algorithm is more general than previously-published algorithms of this form, in that $e$ can be a function of all previous states, actions, and reinforcements, rather than just the current reinforcement. This is what allows VAPS to do both value and policy search.

Every algorithm proposed in this paper is a special case of the VAPS equation on the left side of Table 2. Note that no model is needed for this algorithm. The only probability needed in the algorithm is the policy, not the transition probability from the MDP. This is stochastic gradient descent on $B$, and the update rule is only correct if the observed transitions are sampled from trajectories found by following

Table 2. The general VAPS algorithm (left), and several instantiations of it (right). This single algorithm includes both value-based and policy-search approaches and their combination, and gives guaranteed convergence in every case.

| $\Delta w_t = -\alpha \left[ \frac{\partial}{\partial w} e(s_t) + e(s_t) T_t \right]$ <br><br> $\Delta T_t = \frac{\partial}{\partial w} \ln(P(u_{t-1} \mid s_{t-1}))$ | $e_{SARSA}(s_t) = \frac{1}{2} E^2 \left[ R_{t-1} + \gamma Q(x_t, u_t) - Q(x_{t-1}, u_{t-1}) \right]$ |
|---|---|
| | $e_{Q-learning}(s_t) = \frac{1}{2} E^2 \left[ R_{t-1} + \gamma \max_u Q(x_t, u) - Q(x_{t-1}, u_{t-1}) \right]$ |
| | $e_{advantage}(s_t) = \frac{1}{2} E^2 \left[ \begin{matrix} R_{t-1} + \gamma \max_u A(x_t, u) - \frac{\Delta t}{k} A(x_{t-1}, u_{t-1}) \\ + \left( \frac{\Delta t}{k} - 1 \right) \max_u A(x_{t-1}, u) \end{matrix} \right]$ |
| | $e_{value-iteration}(s_t) = \frac{1}{2} \left[ \max_{u, t} E[R_{t-1} + \gamma V(x_t)] - V(x_{t-1}) \right]^2$ |
| | $e_{SARSA-policy}(s_t) = (1 - \beta)e_{SARSA}(s_t) + \beta(b - \gamma' R_t)$ |

the current, stochastic policy. Both $e$ and $P$ should be smooth functions of $\mathbf{w}$, and for any given $\mathbf{w}$ vector, $e$ should be bounded. The algorithm is simple, but actually generates a large class of different algorithms depending on the choice of $e$ and when the trace is reset to zero. For a single sequence, sampled by following the current policy, the sum of $\Delta w$ along the sequence will give an unbiased estimate of the true gradient, with finite variance. Therefore, during learning, if weight updates are made at the end of each trial, and if the weights stay within a bounded region, and the learning rate approaches zero, then $B$ will converge with probability one. Adding a weight-decay term (a constant times the 2-norm of the weight vector) onto $B$ will prevent weight divergence for small initial learning rates. There is no guarantee that a global minimum will be found when using general function approximators, but at least it will converge. This is true for backprop as well.

## 3 INSTANTIATING THE VAPS ALGORITHM

Many reinforcement-learning algorithms are *value-based*; they try to learn a value function that satisfies the Bellman equation. Examples are Q-learning, which learns a value function, actor-critic algorithms, which learn a value function and the policy which is greedy with respect to it, and TD(1), which learns a value function based on future rewards. Other algorithms are pure *policy-search* algorithms; they directly learn a policy that returns high rewards. These include REINFORCE (Williams, 1988), backprop through time, learning automata, and genetic algorithms. The algorithms proposed here combine the two approaches: they perform *Value And Policy Search* (VAPS). The general VAPS equation is instantiated by choosing an expression for $e$. This can be a Bellman residual (yielding value-based), the reinforcement (yielding policy-search), or a linear combination of the two (yielding Value And Policy Search). The single VAPS update rule on the left side of Table 2 generates a variety of different types of algorithms, some of which are described in the following sections.

### 3.1 REDUCING MEAN SQUARED RESIDUAL PER TRIAL

If the MDP has terminal states, and a *trial* is the time from the start until a terminal state is reached, then it is possible to minimize the expected total error per trial by resetting the trace to zero at the start of each trial. Then, a convergent form of SARSA, $Q$-learning, incremental value iteration, or advantage learning can be generated by choosing $e$ to be the squared Bellman residual, as shown on the right side of Table 2. In each case, the expected value is taken over all possible $(x_t, u_t, R_t)$

triplets, given $s_{t-1}$. The policy must be a smooth, nonzero function of the weights. So it could not be an ε-greedy policy that chooses the greedy action with probability (1-ε) and chooses uniformly otherwise. That would cause a discontinuity in the gradient when two $Q$ values in a state were equal. But the policy could be something that approaches ε-greedy as a positive temperature $c$ approaches zero:

$$P(u \mid x) = \frac{\varepsilon}{n} + (1 - \varepsilon) \frac{1 + e^{Q(x,u)/c}}{\sum_{u'} \left(1 + e^{Q(x,u')/c}\right)} \tag{6}$$

where $n$ is the number of possible actions in each state. For each instance in Table 2 other than value iteration, the gradient of $e$ can be estimated using two, independent, unbiased estimates of the expected value. For example:

$$\frac{\partial}{\partial w} e_{SARSA}(s_t) \doteq e_{SARSA}(s_t)\left( \gamma\phi \frac{\partial}{\partial w} Q(x'_t, u'_t) - \frac{\partial}{\partial w} Q(x_{t-1}, u_{t-1}) \right) \tag{7}$$

When $\phi=1$, this is an estimate of the true gradient. When $\phi<1$, this is a *residual* algorithm, as described in (Baird, 96), and it retains guaranteed convergence, but may learn more quickly than pure gradient descent for some values of $\phi$. Note that the gradient of $Q(x,u)$ at time $t$ uses primed variables. That means a new state and action at time $t$ were generated independently from the state and action at time $t-1$. Of course, if the MDP is deterministic, then the primed variables are the same as the unprimed. If the MDP is nondeterministic but the model is known, then the model must be evaluated one additional time to get the other state. If the model is not known, then there are three choices. First, a model could be learned from past data, and then evaluated to give this independent sample. Second, the issue could be ignored, simply reusing the unprimed variables in place of the primed variables. This may affect the quality of the learned function (depending on how random the MDP is), but doesn't stop convergence, and be an acceptable approximation in practice. Third, all past transitions could be recorded, and the primed variables could be found by searching for all the times $(x_{t-1}, u_{t-1})$ has been seen before, and randomly choosing one of those transitions and using its successor state and action as the primed variables. This is equivalent to learning the certainty equivalence model, and sampling from it, and so is a special case of the first choice. For extremely large state-action spaces with many starting states, this is likely to give the same result in practice as simply reusing the unprimed variables as the primed variables. Note, that when weights do not effect the policy at all, these algorithms reduce to standard residual algorithms (Baird, 95).

It is also possible to reduce the mean squared residual per step, rather than per trial. This is done by making period lengths independent of the policy, so minimizing error per period will also minimize the error per step. For example, a period might be defined to be the first 100 steps, after which the traces are reset, and the state is returned to the start state. Note that if every state-action pair has a positive chance of being seen in the first 100 steps, then this will *not* just be solving a finite-horizon problem. It will be actually be solving the discounted, infinite-horizon problem, by reducing the Bellman residual in every state. But the weighting of the residuals will be determined only by what happens during the first 100 steps. Many different problems can be solved by the VAPS algorithm by instantiating the definition of "period" in different ways.

## 3.2 POLICY-SEARCH AND VALUE-BASED LEARNING

It is also possible to add a term that tries to maximize reinforcement directly. For example, $e$ could be defined to be $e_{SARSA\text{-}policy}$ rather than $e_{SARSA}$ from Table 2, and

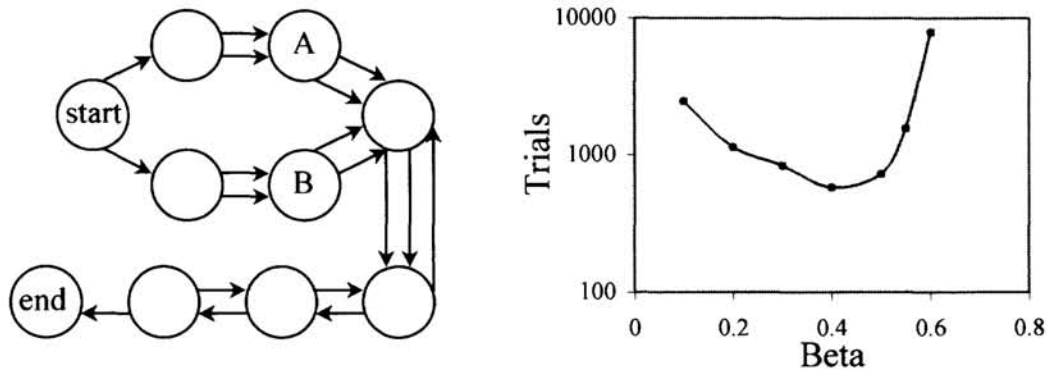

Figure 1. A POMDP and the number of trials needed to learn it vs. $\beta$.
A combination of policy-search and value-based RL outperforms either alone.

the trace reset to zero after each terminal state is reached. The constant $b$ does not affect the expected gradient, but does affect the noise distribution, as discussed in (Williams, 88). When $\beta=0$, the algorithm will try to learn a $Q$ function that satisfies the Bellman equation, just as before. When $\beta=1$, it directly learns a policy that will minimize the expected total discounted reinforcement. The resulting "$Q$ function" may not even be close to containing true $Q$ values or to satisfying the Bellman equation, it will just give a good policy. When $\beta$ is in between, this algorithm tries to both satisfy the Bellman equation and give good greedy policies. A similar modification can be made to any of the algorithms in Table 2. In the special case where $\beta=1$, this algorithm reduces to the REINFORCE algorithm (Williams, 1988). REINFORCE has been rederived for the special case of gaussian action distributions (Tresp & Hofman, 1995), and extensions of it appear in (Marbach, 1998). This case of pure policy search is particularly interesting, because for $\beta=1$, there is no need for any kind of model or of generating two independent successors. Other algorithms have been proposed for finding policies directly, such as those given in (Gullapalli, 92) and the various algorithms from learning automata theory summarized in (Narendra & Thathachar, 89). The VAPS algorithms proposed here appears to be the first one unifying these two approaches to reinforcement learning, finding a value function that both approximates a Bellman-equation solution and directly optimizes the greedy policy.

Figure 1 shows simulation results for the combined algorithm. A run is said to have learned when the greedy policy is optimal for 1000 consecutive trials. The graph shows the average plot of 100 runs, with different initial random weights between $\pm 10^{-6}$. The learning rate was optimized separately for each $\beta$ value. $R=1$ when leaving state $A$, $R=2$ when leaving state $B$ or entering *end*, and $R=0$ otherwise. $\gamma=0.9$. The algorithm used was the modified $Q$-learning from Table 2, with exploration as in equation 13, and $\varphi=c=1$, $b=0$, $\varepsilon=0.1$. States $A$ and $B$ share the same parameters, so ordinary SARSA or greedy $Q$-learning could never converge, as shown in (Gordon, 96). When $\beta=0$ (pure value-based), the new algorithm converges, but of course it cannot learn the optimal policy in the start state, since those two $Q$ values learn to be equal. When $\beta=1$ (pure policy-search), learning converges to optimality, but slowly, since there is no value function caching the results in the long sequence of states near the end. By combining the two approaches, the new algorithm learns much more quickly than either alone.

It is interesting that the VAPS algorithms described in the last three sections can be applied directly to a Partially Observable Markov Decision Process (POMDP), where the true state is hidden, and all that is available on each time step is an

ambiguous "observation", which is a function of the true state. Normally, an algorithm such as SARSA only has guaranteed convergance when applied to an MDP. The VAPS algorithms will converge in such cases.

## 4  CONCLUSION

A new algorithm has been presented. Special cases of it give new algorithms similar to *Q*-learning, SARSA, and advantage learning, but with guaranteed convergence for a wider range of problems than was previously possible, including POMDPs. For the first time, these can be guaranteed to converge, even when the exploration policy changes during learning. Other special cases allow new approaches to reinforcement learning, where there is a tradeoff between satisfying the Bellman equation and improving the greedy policy. For one MDP, simulation showed that this combined algorithm learned more quickly than either approach alone. This unified theory, unifying for the first time both value-based and policy-search reinforcement learning, is of theoretical interest, and also was of practical value for the simulations performed. Future research with this unified framework may be able to empirically or analytically address the old question of when it is better to learn value functions and when it is better to learn the policy directly. It may also shed light on the new question, of when it is best to do both at once.

**Acknowledgments**

This research was sponsored in part by the U.S. Air Force.

**References**

Baird, L. C. (1995). Residual Algorithms: Reinforcement Learning with Function Approximation. In Armand Prieditis & Stuart Russell, eds. Machine Learning: Proceedings of the Twelfth International Conference, 9-12 July, Morgan Kaufman Publishers, San Francisco, CA.

Gordon, G. (1996). "Stable fitted reinforcement learning". In G. Tesauro, M. Mozer, and M. Hasselmo (eds.), Advances in Neural Information Processing Systems 8, pp. 1052-1058. MIT Press, Cambridge, MA.

Gullapalli, V. (1992). *Reinforcement Learning and Its Application to Control.* Dissertation and COINS Technical Report 92-10, University of Massachusetts, Amherst, MA.

Kaelbling, L. P., Littman, M. L. & Cassandra, A., "Planning and Acting in Partially Observable Stochastic Domains". Artificial Intelligence, to appear. Available now at http://www.cs.brown.edu/people/lpk.

Marbach, P. (1998). Simulation-Based Optimization of Markov Decision Processes. Thesis LIDS-TH 2429, Massachusetts Institute of Technology.

McCallum (1995), A. *Reinforcement learning with selective perception and hidden state.* Dissertation, Department of Computer Science, University of Rochester, Rochester, NY.

Narendra, K., & Thathachar, M.A.L. (1989). *Learning automata: An introduction.* Prentice Hall, Englewood Cliffs, NJ.

Tresp, V., & R. Hofman (1995). "Missing and noisy data in nonlinear time-series prediction". In *Proceedings of Neural Networks for Signal Processing 5*, F. Girosi, J. Makhoul, E. Manolakos and E. Wilson, eds., IEEE Signal Processing Society, New York, New York, 1995. pp. 1-10.

Williams, R. J. (1988). *Toward a theory of reinforcement-learning connectionist systems.* Technical report NU-CCS-88-3, Northeastern University, Boston, MA.